# Algebraic Information Geometry for Learning Machines with Singularities

**Sumio Watanabe**

Precision and Intelligence Laboratory
Tokyo Institute of Technology
4259 Nagatsuta, Midori-ku, Yokohama, 226-8503 Japan
*swatanab@pi.titech.ac.jp*

## Abstract

Algebraic geometry is essential to learning theory. In hierarchical learning machines such as layered neural networks and gaussian mixtures, the asymptotic normality does not hold, since Fisher information matrices are singular. In this paper, the rigorous asymptotic form of the stochastic complexity is clarified based on resolution of singularities and two different problems are studied. (1) If the prior is positive, then the stochastic complexity is far smaller than BIC, resulting in the smaller generalization error than regular statistical models, even when the true distribution is not contained in the parametric model. (2) If Jeffreys' prior, which is coordinate free and equal to zero at singularities, is employed then the stochastic complexity has the same form as BIC. It is useful for model selection, but not for generalization.

## 1 Introduction

The Fisher information matrix determines a metric of the set of all parameters of a learning machine [2]. If it is positive definite, then a learning machine can be understood as a Riemannian manifold. However, almost all learning machines such as layered neural networks, gaussian mixtures, and Boltzmann machines have singular Fisher metrics. For example, in a three-layer perceptron, the Fisher information matrix $I(w)$ for a parameter $w$ is singular ($\det I(w) = 0$) if and only if $w$ represents a small model which can be realized with the fewer hidden units than the learning model. Therefore, when the learning machine is in an almost redundant state, any method in statistics and physics that uses a quadratic approximation of the loss function can not be applied. In fact, the maximum likelihood estimator is not subject to the asymptotic normal distribution [4]. The Bayesian posterior probability converges to a distribution which is quite different from the normal one [8]. To construct a mathematical foundation for such learning machines, we clarified the essential relation between algebraic geometry and Bayesian statistics [9,10]. In this

paper, we show that the asymptotic form of the Bayesian stochastic complexity is rigorously obtained by resolution of singularities. The Bayesian method gives powerful tools for both generalization and model selection, however, the appropriate prior for each purpose is quite different.

## 2 Stochastic Complexity

Let $p(x|w)$ be a learning machine, where $x$ is a pair of an input and an output, and $w \in R^d$ is a parameter. We prepare a prior distribution $\varphi(w)$ on $R^d$. Training samples $X^n = (X_1, X_2, ..., X_n)$ are independently taken from the true distribution $q(x)$, which is not contained in $p(x|w)$ in general. The stochastic complexity $F(X^n)$ and its average $F(n)$ are defined by

$$F(X^n) = -\log \int \prod_{i=1}^{n} p(X_i|w)\, \varphi(w) dw$$

and $F(n) = E_{X^n}\{F(X^n)\}$, respectively, where $E_{X^n}\{\cdot\}$ denotes the expectation value overall training sets. The stochastic complexity plays a central role in Bayesian statistics. Firstly, $F(n+1)-F(n)-S$, where $S = -\int q(x)\log q(x)dx$, is equal to the average Kullback distance from $q(x)$ to the Bayes predictive distribution $p(x|X^n)$, which is called the generalization error denoted by $G(n)$. Secondly, $\exp(-F(X^n))$ is in proportion to the posterior probability of the model, hence, the best model is selected by minimization of $F(X^n)$ [7]. And lastly, if the prior distribution has a hyperparameter $\theta$, that is to say, $\varphi(w) = \varphi(w|\theta)$, then it is optimized by minimization of $F(X^n)$ [1].

We define a function $F_0(n)$ using the Kullback distance $H(w)$,

$$F_0(n) = -\log \int \exp(-nH(w))\varphi(w)dw, \quad H(w) = \int q(x)\log\frac{q(x)}{p(x|w)}dx.$$

Then by Jensen's inequality, $F(n) - Sn \le F_0(n)$. Moreover, we assume that $L(x, w) \equiv \log q(x) - \log p(x|w)$ is an analytic function from $w$ to the Hilbert space of all square integrable functions with the measure $q(x)dx$, and that the support of the prior $W = \text{supp } \varphi$ is compact. Then $H(w)$ is an analytic function on $W$, and there exists a constant $c_1 > 0$ such that, for an arbitrary $n$,

$$F_0(\frac{n}{2}) - c_1 \le F(n) - Sn \le F_0(n). \tag{1}$$

## 3 General Learning Machines

In this section, we study a case when the true distribution is contained in the parametric model, that is to say, there exists a parameter $w_0 \in W$ such that $q(x) = p(x|w_0)$. Let us introduce a zeta function $J(z)$ $(z \in C)$ of $H(w)$ and a state density function $v(t)$ by

$$J(z) = \int H(w)^z \varphi(w)dw, \quad v(t) = \int \delta(t - H(w))\varphi(w)dw.$$

Then, $J(z)$ and $F_0(n)$ are represented by the Mellin and the Laplace transform of $v(t)$, respectively.

$$J(z) = \int_0^h t^z v(t)dt, \quad F_0(n) = -\log \int_0^h \exp(-nt)v(t)dt,$$

where $h = \max_{w \in W} H(w)$. Therefore $F_0(n)$, $v(t)$, and $J(z)$ are mathematically connected. It is obvious that $J(z)$ is a holomorphic function in $\mathrm{Re}(z) > 0$. Moreover, by using the existence of Sato-Bernstein's b-function [6], it can be analytically continued to a meromorphic function on the entire complex plane, whose poles are real, negative, and rational numbers. Let $-\lambda_1 > -\lambda_2 > -\lambda_3 > \cdots$ be the poles of $J(z)$ and $m_k$ be the order of $-\lambda_k$. Then, by using the inverse Mellin tansform, it follows that $v(t)$ has an asymptotic expansion with coefficients $\{c_{km}\}$,

$$v(t) \cong \sum_{k=1}^{\infty} \sum_{m=1}^{m_k} c_{km} t^{\lambda_k - 1} (-\log t)^{m-1} \quad (t \to +0).$$

Therefore, also $F_0(n)$ has an asymptotic expansion, by putting $\lambda = \lambda_1$ and $m = m_1$,

$$F_0(n) = \lambda \log n - (m-1) \log \log n + O(1),$$

which ensures the asymptotic expansion of $F(n)$ by eq.(1),

$$F(n) = Sn + \lambda \log n - (m-1) \log \log n + O(1).$$

The Kullback distance $H(w)$ depends on the analytic set $W_0 = \{w \in W; H(w) = 0\}$, resulting that both $\lambda$ and $m$ depend on $W_0$. Note that, if the Bayes generalization error $G(n) = F(n+1) - F(n) - S$ has an asymptotic expansion, it should be $\lambda/n - (m-1)/(n \log n)$. The following lemma is proven using the definition of $F_0(n)$ and its asymptotic expansion.

**Lemma 1** *(1) Let $(\lambda_i, m_i)$ $(i = 1, 2)$ be constants corresponding to $(H_i(w), \varphi_i(w))$ $(i = 1, 2)$. If $H_1(w) \le H_2(w)$ and $\varphi_1(w) \ge \varphi_2(w)$, then '$\lambda_1 < \lambda_2$' or '$\lambda_1 = \lambda_2$ and $m_1 \ge m_2$'.*
*(2) Let $(\lambda_i, m_i)$ $(i = 1, 2)$ be constants corresponding to $(H_i(w_i), \varphi_i(w_i))$ $(i = 1, 2)$. Let $w = (w_1, w_2)$, $H(w) = H_1(w_1) + H_2(w_2)$, and $\varphi(w) = \varphi_1(w_1)\varphi_2(w_2)$. Then the constants of $(H(w), \varphi(w))$ are $\lambda = \lambda_1 + \lambda_2$ and $m = m_1 + m_2 - 1$.*

The concrete values of $\lambda$ and $m$ can be algorithmically obtained by the following theorem. Let $W^i$ be the open kernel of $W$ (the maximal open set contained in $W$).

**Theorem 1** *(Resolution of Singularities, Hironaka [5]) Let $H(w) \ge 0$ be a real analytic function on $W^i$. Then there exist both a real $d$-dimensional manifold $U$ and a real analytic function $g : U \to W^i$ such that, in a neighborhood of an arbitrary $u \in U$,*

$$H(g(u)) = a(u) u_1^{2s_1} u_2^{2s_2} \cdots u_d^{2s_d} \tag{2}$$

*where $a(u) > 0$ is an analytic function and $\{s_i\}$ are non-negative integers. Moreover, for arbitrary compact set $K \subset W$, $g^{-1}(K) \subset U$ is a compact set. Such a function $g(u)$ can be found by finite blowing-ups.*

**Remark.** By applying eq.(2) to the definition of $J(z)$, one can see the integral in $J(z)$ is decomposed into a direct product of the integral of each variable [3]. Applications to learning theory are shown in [9,10]. In general it is not so easy to find $g(u)$ that gives the complete resolution of singularities, however, in this paper, we show that even a partial resolution mapping gives an upper bound of $\lambda$.

**Definition.** We introduce two different priors.
(1) The prior distribution $\varphi(w)$ is called positive if $\varphi(w) > 0$ for an arbitrary

$w \in W^i$, ($W = \mathrm{supp}\varphi(w)$).

(2) The prior distribution $\phi(w)$ is called Jeffreys' one if

$$\phi(w) = \frac{1}{Z}\sqrt{\det I(w)}, \quad I_{ij}(w) = \int \frac{\partial L}{\partial w_i}\frac{\partial L}{\partial w_j}p(x|w)dx,$$

where $Z$ is a normalizing constant and $I(w)$ is the Fisher information matrix. In neural networks and gaussian mixtures, Jeffreys' prior is not positive, since $\det I(w) = 0$ on the parameters which represent the smaller models.

**Theorem 2** *Assume that there exists a parameter $w_0 \in W^i$ such that $q(x) = p(x|w_0)$. Then followings hold.*
*(1) If the prior is positive, then $0 < \lambda \le d/2$ and $1 \le m \le d$. If $p(x|w)$ satisfies the condition of the asymptotic normality, then $\lambda = d/2$ and $m = 1$.*
*(2) If Jeffreys' prior is applied, then '$\lambda > d/2$' or '$\lambda = d/2$ and $m = 1$'.*

(Outline of the Proof) (1) In order to examine the poles of $J(z)$, we can divide the parameter space into the sum of neighborhoods. Since $H(w)$ is an analytic function, in arbitrary neighborhood of $w_0$ that satisfies $H(w_0) = 0$, we can find a positive definite quadratic form which is smaller than $H(w)$. The positive definite quadratic form satisfies $\lambda = d/2$ and $m = 1$. By using Lemma 1 (1), we obtain the first half.
(2) Because Jeffreys' prior is coordinate free, we can study the problem on the parameter space $U$ instead of $W^i$ in eq.(2). Hence, there exists an analytic function $t(x, u)$ such that, in each local coordinate,

$$L(x, u) = L(x, g(u)) = t(x, u)u_1^{s_1}\cdots u_d^{s_d}.$$

For simplicity, we assume that $s_i > 0$ $(i = 1, 2, ..., d)$. Then

$$\frac{\partial L}{\partial w_i} = (\frac{\partial t}{\partial w_i}w_i + s_i t)u_1^{s_1}\cdots u_i^{s_i-1}\cdots u_d^{s_d}.$$

By using blowing-ups $u_i = v_1 v_2\cdots v_i$ $(i = 1, 2, ..., d)$ and a notation $\sigma_p = s_p + s_{p+1} + \cdots + s_d$, it is easy to show

$$\det I(v) \le \prod_{p=1}^{d} v_p^{2d\sigma_p+p-d-2}, \quad du = (\prod_{p=1}^{d}|v_p|^{d-p})dv. \tag{3}$$

By using $H(g(u))^z = \prod_p v_p^{2\sigma_p z}$ and Lemma.1 (1), in order to prove the latter half of the theorem, it is sufficient to prove that

$$\hat{J}(z) \equiv \prod_{p=1}^{d}\int_{|v_p|<h'} u_p^{2\sigma_p z}\cdot|v_p|^{d\sigma_p-1+(d-p)/2}dv_p$$

has a pole $z = -d/2$ with the order $m = 1$. Direct calculation of integrals in $\hat{J}(z)$ completes the theorem. (Q.E.D.)

## 4 Three-Layer Perceptron

In this section, we study some cases when the learner is a three-layer perceptron and the true distribution is contained and not contained. We define the three layer

perceptron $p(x, y|w)$ with $M$ input units, $K$ hidden units, and $N$ output units, where $x$ is an input, $y$ is an output, and $w$ is a parameter.

$$p(x, y|w) = \frac{r(x)}{(2\pi\sigma^2)^{N/2}} \exp(-\frac{1}{2\sigma^2}\|y - f_K(x, w)\|^2)$$

$$f_K(x, w) = \sum_{k=1}^{K} a_k \sigma(b_k \cdot x + c_k)$$

where $w = \{(a_k, b_k, c_k); a_k \in R^N, b_k \in R^M, c_k \in R^1\}$, $r(x)$ is the probability density on the input, and $\sigma^2$ is the variance of the output (either $r(x)$ or $\sigma$ is not estimated).

**Theorem 3** *If the true distribution is represented by the three-layer perceptron with $K_0 \leq K$ hidden units, and if positive prior is employed, then*

$$\lambda \leq \frac{1}{2}\{K_0(M + N + 1) + (K - K_0)\min(M + 1, N)\}. \tag{4}$$

(Outline of Proof) Firstly, we consider a case when $g(x) = 0$. Then,

$$H(w) = \frac{1}{2\sigma^2} \int \{\sum_{k=1}^{K} a_k \tanh(b_k \cdot z) + c_k\}^2 r(x)dx. \tag{5}$$

Let $a_k = (a_{k1}, ..., a_{kN})$ and $b_k = (b_{k1}, ..., b_{kM})$. Let us consider a blowing-up,

$$a_{11} = \alpha, \quad a_{kj} = \alpha a'_{kj} \ (k \neq 1, j \neq 1), \quad b_{kl} = b'_{kl}, \quad c_k = c'_k.$$

Then $da \ db \ dc = \alpha^{KN-1}d\alpha \ da' \ db' \ dc'$ and there exists an analytic function $H_1(a', b', c')$ such that $H(a, b, c) = \alpha^2 H_1(a', b', c')$. Therefore $J(z)$ has a pole at $z = -KN/2$. Also by using another blowing-up,

$$a_{kj} = a''_{kj}, \quad c_1 = \alpha, \quad b_{kl} = \alpha b''_{kl}, \quad c_k = \alpha c''_k \ (k \neq 1),$$

then, $da \ db \ dc = \alpha^{(M+1)K-1}d\alpha \ da'' \ db'' \ dc''$ and there exists an analytic function $H_2(a'', b'', c'')$ such that $H(a, b, c) = \alpha^2 H_2(a'', b'', c'')$, which shows that $J(z)$ has a pole at $z = -K(M + 1)/2$. By combining both results, we obtain $\lambda \leq (K/2)\min(M + 1, N)$. Secondly, we prove the general case, $0 < K_0 \leq K$. Then,

$$H(w) \leq \frac{1}{\sigma^2} \int \{\sum_{k=1}^{K_0} a_k \tanh(b_k \cdot x + c_k) - g(x)\}^2 r(x)dx$$

$$+ \frac{1}{\sigma^2} \int \{\sum_{k=K_0+1}^{K} a_k \tanh(b_k \cdot x + c_k)\}^2 r(x)dx. \tag{6}$$

By combining Lemma.1 (2) and the above result, we obtain the Theorem. (Q.E.D.).

If the true regression function $g(x)$ is not contained in the learning model, we assume that, for each $0 \leq k \leq K$, there exists a parameter $w_0^{(k)} \in W$ that minimizes the square error

$$E_k(w) = \int \|g(x) - f_k(x, w)\|^2 r(x)dx.$$

We use notations $E(k) \equiv E_k(w_0^{(k)})$ and $\lambda(k) = (1/2)\{k(M+N+1) + (K-k)\min(M+1, N)$.

**Theorem 4** *If the true regression function is not contained in the learning model and positive prior is applied, then*

$$F(n) \leq \min_{0 \leq k \leq K} \left[\frac{n}{\sigma^2} E(k) + \lambda(k)\log n\right] + O(1).$$

(Outline of Proof) This theorem can be shown by the same procedure as eq.(6) in the preceding theorem. (Q.E.D.)

If $G(n)$ has an asymptotic expansion $G(n) = \sum_{q=1}^{Q} a_q f_q(n)$, where $f_q(n)$ is a decreasing function of $n$ that satisfies $f_{q+1}(n) = o(f_q(n))$ and $f_Q(n) = 1/n$, then

$$G(n) \leq \min_{0 \leq k \leq K} \left[\frac{E(k)}{\sigma^2} + \frac{\lambda(k)}{n}\right],$$

which shows that the generalization error of the layered network is smaller than the regular statistical models even when the true distribution is not contained in the learning model. It should be emphasized that the optimal $k$ that minimizes $G(n)$ is smaller than the learning model when $n$ is not so large, and it becomes larger as $n$ increases. This fact shows that the positive prior is useful for generalization but not appropriate for model selection. Under the condition that the true distribution is contained in the parametric model, Jeffreys' prior may enable us to find the true model with higher probability.

**Theorem 5** *If the true regression function is contained in the three-layer perceptron and Jeffrey's prior is applied, then $\lambda = d/2$ and $m = 1$, even if the Fisher metric is degenerate at the true parameter.*

(Outline of Proof) For simplicity, we prove the theorem for the case $g(x) = 0$. The general cases can be proven by the same method. By direct calculation of the Fisher information matrix, there exists an analytic function $D(b, c) \geq 0$ such that

$$\det I(w) = \prod_{k=1}^{K}\left(\sum_{p=1}^{N} a_{kp}\right)^{2(M+1)} D(b, c)$$

By using a blowing-up

$$a_{11} = \alpha, \quad a_{kj} = \alpha a'_{kj} \ (k \neq 1, j \neq 1), \quad b_{kl} = b'_{kl}, \quad c_k = c'_k,$$

we obtain $H(w) = \alpha^2 H_1(a', b', c')$ same as eq.(5), $\det I(w) \propto \alpha^{2(M+1)K}$, and $da\,db\,dc = \alpha^{NK-1} d\alpha\,da'\,db\,dc$. The integral

$$\hat{J}(z) = \int_{|\alpha| < \epsilon'} \alpha^{2z} \alpha^{(M+1)K+NK-1} d\alpha$$

has a pole at $z = -(M+N+1)K/2$. By combining this result with Theorem 3, we obtain Theorem.5. (Q.E.D.).

# 5 Discussion

In many applications of neural networks, rather complex machines are employed compared with the number of training samples. In such cases, the set of optimal parameters is not one point but an analytic set with singularities, and the set of almost optimal parameters $\{w; H(w) < \epsilon\}$ is not an 'ellipsoid'. Hence neither the Kullback distance can be approximated by any quadratic form nor the saddle point approximation can be used in integration on the parameter space. The zeta function of the Kullback distance clarifies the behavior of the stochastic complexity and resolution of singularities enables us to calculate the learning efficiency.

# 6 Conclusion

The relation between algebraic geometry and learning theory is clarified, and two different facts are proven.
(1) If the true distribution is not contained in a hierarchical learning model, then by using a positive prior, the generalization error is made smaller than the regular statistical models.
(2) If the true distribution is contained in the learning model and if Jeffreys' prior is used, then the average Bayesian factor has the same form as BIC.

### Acknowledgments

This research was partially supported by the Ministry of Education, Science, Sports and Culture in Japan, Grant-in-Aid for Scientific Research 12680370.

### References

[1] Akaike, H. (1980) Likelihood and Bayes procedure. *Bayesian Statistics*, (Bernald J.M. eds.) University Press, Valencia, Spain, 143-166.

[2] Amari, S. (1985) *Differential-geometrical methods in Statistics*. Lecture Notes in Statistics, Springer.

[3] Atiyah, M. F. (1970) Resolution of singularities and division of distributions. *Comm. Pure and Appl. Math.* , 13, pp.145-150.

[4] Dacunha-Castelle, D., & Gassiat, E. (1997). Testing in locally conic models, and application to mixture models. *Probability and Statistics*, 1, 285-317.

[5] Hironaka, H. (1964) Resolution of Singularities of an algebraic variety over a field of characteristic zero. *Annals of Math.*, 79,109-326.

[6] Kashiwara, M. (1976) B-functions and holonomic systems. *Inventions Math.*, 38,33-53.

[7] Schwarz, G. (1978) Estimating the dimension of a model. *Ann. of Stat.*, 6 (2), 461-464.

[8] Watanabe, S. (1998) On the generalization error by a layered statistical model with Bayesian estimation. *IEICE Transactions*, J81-A (10), 1442-1452. English version: (2000)*Electronics and Communications in Japan*, Part 3, 83(6) ,95-104.

[9] Watanabe, S. (2000) Algebraic analysis for non-regular learning machines. *Advances in Neural Information Processing Systems*, 12, 356-362.

[10] Watanabe, S. (2001) Algebraic analysis for non-identifiable learning machines. *Neural Computation*, to appear.
